# Combining Visual and Acoustic Speech Signals with a Neural Network Improves Intelligibility

**T.J. Sejnowski**
The Salk Institute
and
Department of Biology
The University of
California at San Diego
San Diego, CA 92037

**B.P. Yuhas**
**M.H. Goldstein, Jr.**
Department of Electrical
and Computer
Engineering
The Johns Hopkins
University
Baltimore, MD 21218

**R.E. Jenkins**
The Applied Physics
Laboratory
The Johns Hopkins
University
Laurel, MD 20707

## ABSTRACT

Acoustic speech recognition degrades in the presence of noise. Compensatory information is available from the visual speech signals around the speaker's mouth. Previous attempts at using these visual speech signals to improve automatic speech recognition systems have combined the acoustic and visual speech information at a symbolic level using heuristic rules. In this paper, we demonstrate an alternative approach to fusing the visual and acoustic speech information by training feedforward neural networks to map the visual signal onto the corresponding short-term spectral amplitude envelope (STSAE) of the acoustic signal. This information can be directly combined with the degraded acoustic STSAE. Significant improvements are demonstrated in vowel recognition from noise-degraded acoustic signals. These results are compared to the performance of humans, as well as other pattern matching and estimation algorithms.

## 1   INTRODUCTION

Current automatic speech recognition systems rely almost exclusively on the acoustic speech signal, and as a consequence, these systems often perform poorly in noisy

environments. To compensate for noise-degradation of the acoustic signal, one can either attempt to remove the noise from the acoustic signal or supplement the acoustic signal with other sources of speech information. One such source is the visible movements of the mouth. For humans, visual speech signals can improve speech perception when the acoustic signal is degraded by noise (Sumby and Pollack, 1954) and can serve as a source of speech information when the acoustic signal is completely absent through lipreading. How can these visual speech signals be used to improve the automatic recognition of speech?

One speech recognition system that has extensively used the visual speech signals was developed by Eric Petajan (1987). For a limited vocabulary, Petajan demonstrated that the visual speech signals can be used to significantly improve automatic speech recognition compared to the acoustic recognition alone. The system relied upon a codebook of images that were used to translate incoming images into corresponding symbols. These symbol strings were then compared to stored sequences representing different words in the vocabulary. This categorical treatment of speech signals is required because of the computational limitations of currently available digital serial hardware.

This paper proposes an alternative method for processing visual speech signals based on analog computation in a distributed network architecture. By using many interconnected processors working in parallel large amounts of data can be handled concurrently. In addition to speeding up the computation, this approach does not require segmentation in the early stages of processing; rather, analog signals from the visual and auditory pathways flow through networks in real time and can be combined directly.

Results are presented from a series of experiments that use neural networks to process the visual speech signals of two talkers. In these preliminary experiments, the results are limited to static images of vowels. We demonstrate that these networks are able to extract speech information from the visual images, and that this information can be used to improve automatic vowel recognition.

## 2    VISUAL AND ACOUSTIC SPEECH SIGNALS

The acoustic speech signal can be modeled as the response of the vocal tract filter to a sound source (Fant, 1960). The resonances of the vocal tract are called *formants*. They often appear as peaks in the short-term power spectrum, and are sufficient to identify the individual vowels (Peterson and Barney, 1953). The overall shape of the short-time spectra is important for general speech perception (Cole, 1980).

The configuration of the articulators define the shape of the vocal tract and the corresponding resonance characteristics of the filter. While some of the articulators are visible on the face of the speaker (e.g., the lips, teeth and sometimes the tip of the tongue), others are not. The contribution of the visible articulators to the acoustic signal results in speech sounds that are much more susceptible to acoustic noise distortion than are the contributions from the hidden articulators (Petajan, 1987), and therefore, the visual speech signal tends to complement the acoustic

signal. For example, the visibly distinct speech sounds /b/ and /k/ are among the first pairs to be confused when presented acoustically in the presence of noise. Because of this complementary structure, the perception of speech in noise is greatly improved when both speech signals are present. How and at what level are these two speech signals being combined?

In previous attempts at using the visual speech signals, the information from the visual signal was incorporated into the recognition system after the signals were categorized (Petajan, 1987). In the approach taken here, visual signals will be used to resolve ambiguities in the acoustic signal before either is categorized. By combining these two sources of information at an early stage of processing, it is possible to reduce the number of erroneous decisions made and increase the amount of information passed to later stages of processing (Summerfield, 1987). The additional information provided by the visual signal can serve to constrain the possible interpretations of an ambiguous acoustic signal, or it can serve as an alternative source of speech information when the acoustical signal is heavily noise-corrupted. In either case, a massive amount of computation must be performed on the raw data. New massively-parallel architectures based on neural networks and new training procedures have made this approach feasible.

## 3   INTERPRETING THE VISUAL SIGNALS

In our approach, the visual signal was mapped directly into an acoustic representation closely related to the vocal tract's transfer function (Summerfield, 1987). This representation allowed the visual signal to be fused with the acoustic signal prior to any symbolic encoding.

The visual signals provide only a partial description of the vocal tract transfer function and that description is usually ambiguous. For a given visual signal there are many possible configurations of the full vocal tract, and consequently many possible corresponding acoustic signals. The goal was to define a *good* estimate of that acoustic signal from the visual signal and then use that estimate in conjunction with any residual acoustic information.

The speech signals used in these experiments were obtained from a male speaker who was video taped while seated facing the camera, under well-lit conditions. The visual and acoustic signals were then transferred and stored on laser disc (Bernstein and Eberhardt, 1986), which allowed the access of individual video frames and the corresponding sound track. The NTSC video standard is based upon 30 frames per second and words are preserved as a series of frames on the laser disc. A data set was constructed of 12 examples of 9 different vowels (Yuhas *et al.*, 1989).

A reduced area-of-interest in the image was automatically defined and centered around the mouth. The resulting sub-image was sampled to produce a topographically accurate image of 20 x 25 pixels that would serve to represent the visual speech signal. While not the most efficient encoding one could use, it is faithful to the parallel approach to computation advocated here and represents what one might observe through an array of sensors.

Along with each video frame on the laser disc there is 33 ms of acoustic speech. The representation chosen for the acoustic output structure was the short-time spectral amplitude envelope (STSAE) of the acoustic signal, because it is essential to speech recognition and also closely related to the vocal tract's transfer function. It can be calculated from the short-term power spectrum of the acoustic signal. The speech signal was sampled and cepstral analysis was used to produced a smooth envelope of the original power spectrum that could be sampled at 32 frequencies.

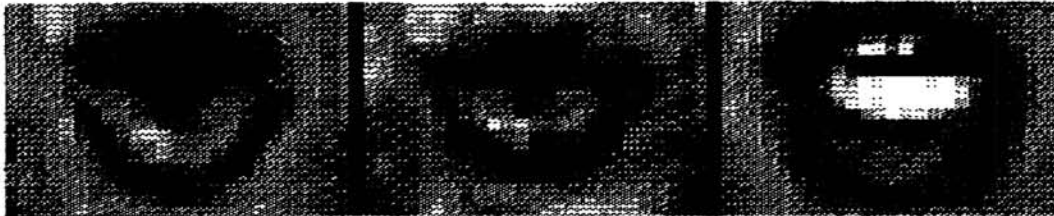

**Figure 1:** Typical lip images presented to the network.

Three-layered feedforward networks with non-linear units were used to perform the mapping. A lip image was presented across 500 input units, and an estimated STSAE was produced across 32 output units. Networks with five hidden units were found to provide the necessary bandwidth while minimizing the effects of over-learning. The standard backpropagation technique was used to compute the error gradients for training the network. However, instead of using a fixed-step steepest-descent algorithm for updating the weights, the error gradient was used in a conjugate-gradient algorithm. The weights were changed only after all of the training patterns were presented.

## 4    INTEGRATING THE VISUAL AND ACOUSTIC SPEECH SIGNALS

To evaluate the spectral estimates, a feedforward network was trained to recognize vowels from their STSAE's. With no noise present, the trained network could correctly categorized 100% of the 54 STSAE's in its training set: thus serving as a *perfect* recognizer for this data. The vowel recognizer was then presented with speech information through two channels, as shown in Fig. 2. The path on the bottom represents the information obtained from the acoustic signal, while the path on the top provides information obtained from the corresponding visual speech signal.

To assess the performance of the recognizer in noise, clean spectral envelopes were systematically degraded by noise and then presented to the recognizer. In this particular condition, no visual input was given to the network. The noise was introduced by adding a normalized random vector to the STSAE. Noise corrupted vectors were produced at 3 dB intervals from -12 dB to 24 dB. At each step 6 different vectors were produced, and the performance reported was the average. Fig. 3 shows the recognition rates as a function of the speech-to- noise ratio. At a speech-to-noise ratio of -12 dB, the recognizer was operating at chance or 11.1%.

Next, a network trained to estimate the spectral envelopes from images was used

to provide an independent STSAE input into the recognizer (along the top of Fig. 2). This network was not trained on any of the data that was used in training the vowel recognizer. The task remained to combine these two STSAE's.

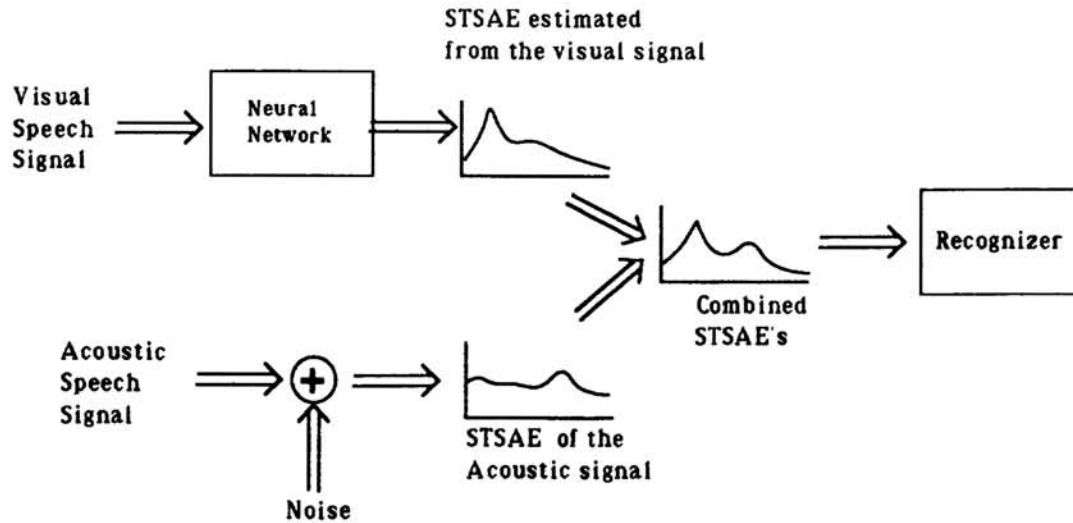

**Figure 2:** A vowel recognizer that integrates the acoustic and visual speech signals.

We considered three different ways of combining the estimates obtained from visual signals with the noised degraded acoustic envelopes. The first approach was to simply average the two envelopes, which proved to be less than optimal. The recognizer was able to identify 55.6% of the STSAE estimated from the visual signal, but when the visual estimate was combined with the noise degraded acoustic signal the recognizer was only capable of 35% at a S/N of -12 dB. Similarly, at very high signal-to-noise ratios, the combined input produced poorer results than the acoustic signal alone provided. To correct for this, the two inputs needed to be weighted according to the relative amount of information available from each source. A weighting factor was introduced which was a function of speech-to-noise:

$$\alpha \, S_{Visual} \; + \; (1 - \alpha) \, S_{Acoustic} \tag{1}$$

The optimal value for the parameter $\alpha$ was found empirically to vary linearly with the speech-to-noise ratio in dB. The value for $\alpha$ ranged from approximately 0.8 at S/N of -12dB to 0.0 at 24 dB. The results obtained from using the $\alpha$ weighted average are shown in Fig. 3.

The third method used to fuse the two STSAE's was with a second-order neural network (Rumelhart *et al.* 1986). Sigma-pi networks were trained to take in noise-degraded acoustic envelopes and estimated envelopes from the corresponding visual speech signal. The networks were able to recreate the noise-free acoustic envelope with greater accuracy than any of the other methods, as measured by mean squared error. This increased accuracy did not however translate into improved recognition rates.

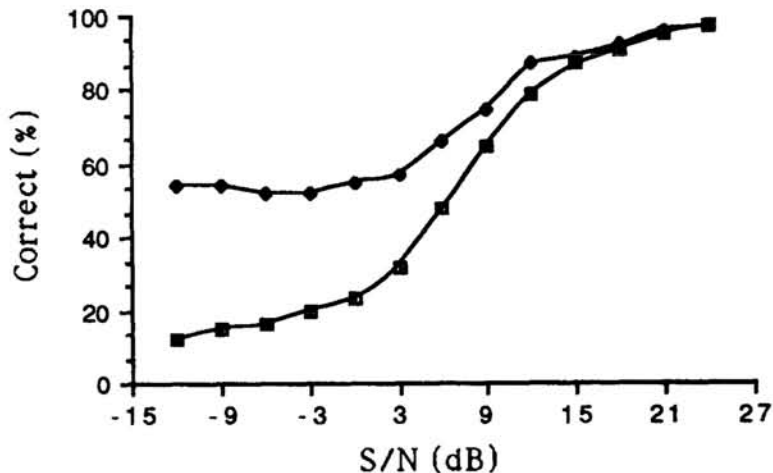

**Figure 3:** The visual contribution to speech recognition in noise. The lower curve shows the performance of the recognizer under varying signal-to-noise conditions using only the acoustic channel. The top curve shows the final improvement when the two channels were combined using the $\alpha$ weighted average.

## 5   COMPARING PERFORMANCE

The performance of the network was compared to more traditional signal-processing techniques.

### 5.1   K-NEAREST NEIGHBORS

In this first comparison, an estimate of the STSAE was obtained using a k-nearest neighbors approach. The images in the training set were stored along with their corresponding STSAE calculated from the acoustic signal. These images served as the data base of stored templates. Individual images from the test set were correlated against all of the stored images and the closest $k$ images were selected. The acoustic STSAE corresponding to the $k$ selected images were then averaged to produce an estimate of the STSAE corresponding to the test image. Using this procedure for various values of $k$, average MSE was calculated for the test set. This procedure was then repeated with the test and training set reversed.

For values of k between 2 and 6 the k-nearest neighbor estimator was able to produce STSAE estimates with approximately the same accuracy as the neural networks. Those networks evaluated after 500 training epochs produced estimates with 9% more error than the KNN approach, while those weights corresponding to the networks' *best* performance, as defined above, produced estimates with 5% less error.

#### 5.1.1   PRINCIPAL COMPONENT ANALYSIS

A second method of comparison was to obtain an STSAE estimate using a combination of optimal linear techniques. The first step was to encode the images using a Hotelling transform, which produces an optimal encoding of an image with respect to a least-mean-squared error. The encoded image $y_i$ was computed from the

normalized image $x_i$ using

$$y_i = A(x_i - m_x) \qquad (2)$$

where $m_x$ was the mean image. $A$ was a transformation matrix whose rows were the five largest eigenvectors of the covariance matrix of the images. The vector $y_i$ represents the image as do the hidden units of the neural network.

The second step was to find a mapping from the encoded image vector $y_i$ to the corresponding short-term spectral envelope $s_i$ using a linear least-squares fit. For the $y_i$'s calculated above, a $B$ was found that provided the best estimate of the desired $s_i$:

$$s_i = By_i \qquad (3)$$

If we think of the matrix $A$ as corresponding to the weights from the input layer to the hidden units, then $B$ maps the hidden units to the output units.

The networks trained to produce STSAE estimates were far superior to those obtained using the coefficients of $A$ and $B$. This was true not only for the training data from which $A$ and $B$ were calculated, but also for the test data set. When compared to networks trained for 500 epochs, the networks produced estimates of the STSAE's that were 46% better on the training set and 12% better on the test set.

## 6   CONCLUSION

Humans are capable of combining information received through distinct sensory channels with great speed and ease. The combined use of the visual and acoustic speech signals is just one example of integrating information across modalities. Sumby and Pollack (1954) have shown that the relative improvement provided by the visual signal varies with the signal-to-noise ratio of the acoustic signal. By combining the speech information available from the two speech signals before categorizing, we obtained performance that was comparable to that demonstrated by humans.

We have shown that visual and acoustic speech information can be effectively fused without requiring categorical preprocessing. The low-level integration of the two speech signals was particularly useful when the signal-to-noise ratio ranged from 3 dB to 15 dB, where the combined signals were recognized with a greater accuracy than either of the two component signals alone. In contrast, an independent categorical decisions on each channel would have required additional information in the form of *ad hoc* rules to produce the same level of performance.

Lip reading research has traditionally focused on the identification and evaluation of visual features (Montgomery and Jackson, 1983). Reducing the original speech signals to a finite set of predefined parameters or to discrete symbols can waste a tremendous amount of information. For an automatic recognition system this information may prove to be useful at a later stage of processing. In our approach, speech information in the visual signal is accessed without requiring discrete feature analysis or making categorical decisions.

This line of research has consequences for other problems, such as target identification based on multiple sensors. For example, the same problems arise in designing systems that combine radar and infrared data. Mapping into a common representation using neural network models could also be applied to these problem domains. The key insight is to combine this information at a stage prior to categorization. Neural network learning procedures allow systems to be constructed for performing the mappings as long as sufficient data are available to train the network.

## Acknowledgements

This research was supported by grant AFOSR-86-0256 from the Air Force Office of Scientific Research and by the Applied Physics Laboratory's IRAD.

## References

Bernstein, L.E. and Eberhardt, S.P. (1986). *Johns Hopkins Lipreading Corpus I-II*, Johns Hopkins University, Baltimore, MD

Cole, R.A. (1980). (Ed.) *Perception and Production of Fluent Speech*, Lawrence Erlbaum Assoc, Publishers, Hillsdale, NJ

Fant, G. (1960). *Acoustic Theory of Speech Production.* Mouton & Co., Publishers, The Hague, Netherlands

Montgomery, A. and Jackson, P.L. (1983). Physical Characteristics of the lips underlying vowel lipreading. J. Acoust. Soc. Am. *73*, 2134-2144.

Petajan, E.D. (1987). An improved Automatic Lipreading System To Enhance Speech Recognition. *Bell Laboratories Technical Report No. 11251-871012-111TM.*

Peterson, G.E. and Barney, H.L. (1952). Control Methods Used in a Study of the Vowels. J. Acoust. Soc. Am. *24*, 175-184.

Rumelhart, D.E., Hinton, G.E. and Williams, R.J. (1986). Learning internal representations by error propagation. In: D.E. Rumelhart and J.L. McClelland. (Eds.) *Parallel Distributed Processing in the Microstructure of Cognition: Vol 1. Foundations* MIT Press, Cambridge, MA

Sumby, W.H. and Pollack, I. (1954). Visual Contribution to Speech Intelligibility in Noise. J. Acoust. Soc. Am. *26*, 212-215.

Summerfield, Q.(1987). Some preliminaries to a comprehensive account of audio-visual speech perception. In: B. Dodd and R. Campbell (Eds.) *Hearing by Eye: The Pschology of Lip-Reading*, Lawrence-Erlbaum Assoc, Hillsdale, NJ.

Yuhas, B.P., Goldstein, M.H. Jr. and Sejnowski, T.J. (1989). Integration of Acoustic and Visual Speech Signals Using Neural Networks. IEEE Comm Magazine *27*, November 65-71.